# PG-means: learning the number of clusters in data

**Yu Feng**        **Greg Hamerly**
Computer Science Department
Baylor University
Waco, Texas 76798
{yu_feng, greg_hamerly}@baylor.edu

## Abstract

We present a novel algorithm called PG-means which is able to learn the number of clusters in a classical Gaussian mixture model. Our method is robust and efficient; it uses statistical hypothesis tests on one-dimensional projections of the data and model to determine if the examples are well represented by the model. In so doing, we are applying a statistical test for the entire model at once, not just on a per-cluster basis. We show that our method works well in difficult cases such as non-Gaussian data, overlapping clusters, eccentric clusters, high dimension, and many true clusters. Further, our new method provides a much more stable estimate of the number of clusters than existing methods.

## 1 Introduction

The task of data clustering is important in many fields such as artificial intelligence, data mining, data compression, computer vision, and others. Many different clustering algorithms have been developed. However, most of them require that the user know the number of clusters ($k$) beforehand, while an appropriate value for $k$ is not always clear. It is best to choose $k$ based on prior knowledge about the data, but this information is often not available. Without prior knowledge it can be especially difficult to choose $k$ when the data have high dimension, making exploratory data analysis difficult.

In this paper, we present an algorithm called PG-means (PG stands for projected Gaussian) which is able to discover an appropriate number of Gaussian clusters and their locations and orientations. Our method is a wrapper around the standard and widely used Gaussian mixture model. The paper's primary contribution is a novel method of determining if a whole mixture model fits its data well, based on projections and statistical tests. We show that the new approach works well not only in simple cases in which the clusters are well separated, but also in the situations where the clusters are overlapped, eccentric, in high dimension, and even non-Gaussian. We show that where some other methods tend to severely overfit, our method does not, and that our method is comparable to but much faster than a recent variational Bayes-based approach for learning $k$.

## 2 Related work

Several algorithms have been proposed to determine $k$ automatically. Most of these algorithms wrap around either $k$-means or Expectation-Maximization for fixed $k$. As they proceed, they use splitting or merging rules to increase or decrease $k$ until a proper value is reached.

Pelleg and Moore [9] proposed the X-means algorithm, which is a regularization framework for learning $k$ with $k$-means. This algorithm tries many values for $k$ and obtains a model for each $k$ value. Then X-means uses the Bayesian Information Criterion (BIC) to score each model [5, 12], and chooses the model with the highest BIC score. Besides the BIC, other scoring criteria could also

be applied such as the Akaike Information Criterion [1], or Minimum Description Length [10]. One drawback of the X-means algorithm is that the cluster covariances are all assumed to be spherical and the same width. This can cause X-means to overfit when it encounters data that arise from non-spherical clusters.

Hamerly and Elkan [4] proposed the G-means algorithm, a wrapper around the $k$-means algorithm. G-means uses projection and a statistical test for the hypothesis that the data in a cluster come from a Gaussian distribution. The algorithm grows $k$ starting with a small number of centers. It applies a statistical test to each cluster and those which are not accepted as Gaussians are split into two clusters. Interleaved with $k$-means, this procedure repeats until every cluster's data are accepted as Gaussian. While this method does not assume spherical clusters and works well if true clusters is well-separated, it has difficulties when true clusters overlap since the hard assignment of $k$-means can clip data into subsets that look non-Gaussian.

Sand and Moore [11] proposed an approach based on repairing faults in a Gaussian mixture model. Their approach modifies the learned model at the regions where the residual is large between the model's predicted density and the empirical density. Each modification adds or removes a cluster center. They use a hill-climbing algorithm to seek a model which maximizes a model fitness scoring function. However, calculating the empirical density and comparing it to the model density is difficult, especially in high dimension.

Tibshirani et al. [13] proposed the Gap statistic, which compares the likelihood of a learned model with the distribution of the likelihood of models trained on data drawn from a null distribution. Our experience has shown that this method works well for finding a small number of clusters, but has difficulty as the true $k$ increases.

Welling and Kurihara [15] proposed Bayesian $k$-means, which uses Maximization-Expectation (ME) to learn a mixture model. ME maximizes over the hidden variables (assignment of examples to clusters), and computes an expectation over model parameters (center locations and covariances). It is a special case of variational Bayesian methods. Bayesian $k$-means works well but is slower than our method.

None of these prior approaches perform well in all situations; they tend to overfit, underfit, or are too computationally costly. These issues form the motivation for our new approach.

## 3 Methodology

Our approach is called PG-means, where PG stands for projected Gaussian and refers to the fact that the method applies projections to the clustering model as well as the data, before performing each hypothesis test for model fitness. PG-means uses the standard Gaussian mixture model with Expectation-Maximization training, but any underlying algorithm for training a Gaussian mixture might be used. Our algorithm starts with a simple model and increases $k$ by one at each iteration until it finds a model that fits the data well.

Each iteration of PG-means uses the EM algorithm to learn a model containing $k$ centers. Each time EM learning converges, PG-means projects both the dataset and the learned model to one dimension, and then applies the Kolmogorov-Smirnov (KS) test to determine whether the projected model fits the projected data. PG-means repeats this projection and test step several times for a single learned model. If any test rejects the null hypothesis that the data follows the model's distribution, then it adds one cluster and starts again with EM learning. If every test accepts the null hypothesis for a given model, then the algorithm terminates. Algorithm 1 describes the algorithm more formally.

When adding a new cluster PG-means preserves the $k$ clusters it has learned and adds a new cluster. This preservation helps EM converge more quickly on the new model. To find the best new model, PG-means runs EM 10 times each time it adds a cluster with a different initial location for the new cluster. The mean of each new cluster is chosen from a set of randomly chosen examples, and also points with low model-assigned probability density. The initial covariance of the new cluster is based on the average of the existing clusters' covariances, and the new cluster prior is assigned $1/k$ and all priors are re-normalized. More than 10 EM applications could be used, as well as deterministic annealing [14], to ensure finding the best new model. In our tests, deterministic annealing did not improve the results of PG-means. As stated earlier, any training algorithm (not just EM) may be

---

**Algorithm 1** PG-means (dataset $X$, confidence $\alpha$, number of projections $p$)

---

1: Let $k \leftarrow 1$. Initialize the cluster with the mean and covariance of $X$.
2: **for** $i = 1 \ldots p$ **do**
3:     Project $X$ and the model to one dimension with the same projection.
4:     Use the KS test at significance level $\alpha$ to test if the projected model fits the projected dataset.
5:     If the test rejects the null hypothesis, then break out of the loop.
6: **end for**
7: **if** any test rejected the null hypothesis **then**
8:     **for** $i = 1 \ldots 10$ **do**
9:         Initialize $k + 1$ clusters as the $k$ previously learned plus one new cluster.
10:         Run EM on the $k + 1$ clusters.
11:     **end for**
12:     Retain the model of $k + 1$ clusters with the best likelihood.
13:     Let $k \leftarrow k + 1$, and go to step 2.
14: **end if**
15: Every test accepts the null hypothesis; stop and return the model.

---

used to fit a particular set of $k$ Gaussian models. For example, one might use $k$-means if more speed is desired.

## 3.1 Projection of the model and the dataset

PG-means is novel because it applies projection to the learned model as well as to the dataset prior to testing for model fitness. There are several reasons to project both the examples and the model. First, a mixture of Gaussians remains a mixture of Gaussians after being linearly projected. Second, there are many effective and efficient tests for model fitness in one dimension, but in higher dimensions such testing is more difficult.

Assume some data $X$ is sampled from a single Gaussian cluster with distribution $X \sim N(\mu, \Sigma)$ in $d$ dimensions. So $\mu = E[X]$ is the $d \times 1$ mean vector and $\Sigma = \text{Cov}[X]$ is the $d \times d$ covariance matrix. Given a $d \times 1$ projection vector $P$ of unit length ($||P|| = 1$), we can project $X$ along $P$ as $X' = P^T X$. Then $X' \sim N(\mu', \sigma^2)$, where $\mu' = P^T \mu$ and $\sigma^2 = P^T \Sigma P$. We can project each cluster model to obtain a one-dimensional projection of an entire mixture along $P$. Then we wish to test whether the projected model fits the projected data.

The G-means and X-means algorithms both perform statistical tests for each cluster individually. This makes sense because each algorithm is a wrapper around $k$-means, and $k$-means uses hard assignment (each example has membership in only one cluster). However, this approach is problematic when clusters overlap, since the hard assignment results in 'clipped' clusters, making them appear very non-Gaussian. PG-means tests all clusters and all data at once. Then if two true clusters overlap, the additive probability of the learned Gaussians representing those clusters will correctly model the increased density in the overlapping region.

## 3.2 The Kolmogorov-Smirnov test and critical values

After projection, PG-means uses the univariate Kolmogorov-Smirnov [7] test for model fitness. The KS test statistic is $D = \max_X |F(X) - S(X)|$ – the maximum absolute difference between the true CDF $F(X)$ with the sample CDF $S(X)$. The KS test is only applicable if $F(X)$ is fully specified; however, PG-means estimates the model with EM, so $F(X)$ cannot be specified a priori. The best we can do is use the parameter estimates, but this will cause us to accept the model too readily. In other words, the probability of a Type I error will be too low and PG-means will tend to choose models with too few clusters. Lilliefors [6] gave a table of smaller critical values for the KS test which correct for estimated parameters of a single univariate Gaussian. These values come from Monte Carlo calculations. Along this vein, we create our own test critical values for a mixture of univariate Gaussians.

To generate the critical values for the KS test statistic, we use the projected one-dimensional model that has been learned to generate many different datasets, and then measure the KS test statistic for each dataset. Then we find the KS test statistic that is in the $\alpha$ range we desire, which is the critical value we want. Fortunately, this can be done efficiently and does not dominate the running

time of our algorithm. It is much more efficient than if we were to generate datasets from the full dimensional data and project these to obtain the statistic distribution, yet they are equivalent approaches. Further optimization is possible when we follow Lilliefors' observation that the critical value decreases as approximately $\sqrt{n}$, for sufficiently large $n$, which we have also observed in our simulations with mixtures of Gaussians. Therefore, we can use Monte Carlo simulations with $n' \ll n$ points, and scale the chosen critical value by $\sqrt{n'/n}$. A more accurate scaling given by Dallal and Wilkinson [2] did not offer additional benefit in our tests. We use at most $n' = 3/\alpha$, which is 3000 points for $\alpha = 0.001$. The Monte Carlo simulations can be easily parallelized, and our implementation uses two computational threads.

### 3.3  Number of projections

We wish to use a small but sufficient number of projections and tests to discover when a model does not fit data well. Each projection provides a different view of model fitness along that projection's direction. However, a projection can cause the data from two or more true clusters to be collapsed together, so that the test cannot see that there should be multiple densities used to model them. Therefore multiple projections are necessary to see these model and data discrepancies.

We can choose the projections in several different ways. Random projection [3] provides a useful framework, which is what we use in this paper. Other possible methods include using the leading directions from principal components analysis, which gives a stable set of vectors which can be re-used, or choosing $k - 1$ vectors that span the same subspace spanned by the $k$ cluster centers.

Consider two cluster centers $\mu_1$ and $\mu_2$ in $d$ dimensions and the vector which connects them, $m = \mu_2 - \mu_1$. We assume for simplicity that the two clusters have the same spherical covariance $\Sigma$ and are $c$-separated, that is, $||m|| \geq c\sqrt{\text{trace}(\Sigma)}$. We follow Dasgupta's conclusion that $c$-separation is the natural measure for Gaussians [3]. Now consider the projection of $m$ along some randomly chosen vector $P \sim N(0, 1/d\mathbf{I})$. We use this distribution because in high dimension $P$ will be approximately unit-length. The probability that $P$ is a 'good' projection, i.e. that it maintains $c$-separation between the cluster means when projected, is

$$Pr\left(|P^T m| \geq c\sqrt{P^T \Sigma P}\right) > 1 - \text{Erf}\left(c\sqrt{\frac{dP^T \Sigma P}{2c^2 \text{trace}(\Sigma)}}\right) = 1 - \text{Erf}\left(\sqrt{1/2}\right)$$

where Erf is the standard Gaussian error function. Here we have used the relation $P^T \Sigma P = \text{trace}(\Sigma)/d$ when $\Sigma$ is spherical and $||P|| = 1$. If $\Sigma$ is not spherical, then this is true in an expected sense, i.e. $E[P^T \Sigma P] = \text{trace}(\Sigma)/d$ when $||P|| = 1$. If we perform $p$ random projections, we wish that the probability that all $p$ projections are 'bad' to be less than some $\varepsilon$:

$$Pr(p \text{ bad projections}) = \text{Erf}\left(\sqrt{1/2}\right)^p < \varepsilon$$

Therefore we need approximately $p < \log(\varepsilon)/\log(\text{Erf}(\sqrt{1/2})) \approx -2.6198\log(\varepsilon)$ projections to find a projection that keeps the two cluster means $c$-separated. For $\varepsilon = 0.01$, this is only 12 projections, and for $\varepsilon = 0.001$, this is only 18 projections.

### 3.4  Algorithm complexity

PG-means converges as fast as EM on any given $k$, and it repeats EM every time it adds a cluster. Let $K$ be the final learned number of clusters on $n$ data points. PG-means runs in $O(K^2 n d^2 l + Kn\log(n))$ time, where $l$ is the number of iterations required for EM convergence. The $n\log(n)$ term comes from the sort required for each KS test, and the $d^2$ comes from using full covariance matrices. PG-means uses a fixed number of projections for each model and each projection is linear in $n$, $d$, and $k$; therefore the projections do not increase the algorithm's asymptotic run time. Note also that EM starts with $k$ learned centers and one new randomly initialized center, so EM convergence is much faster in practice than if all $k + 1$ clusters were randomly initialized. We must also factor in the cost of the Monte Carlo simulations for determining the KS test critical value, which are $O(Kd^2 n\log(n)/\alpha)$ for each simulation. For fixed alpha, this does not increase the run-time significantly, and in practice the simulations are a minor part of the running time.

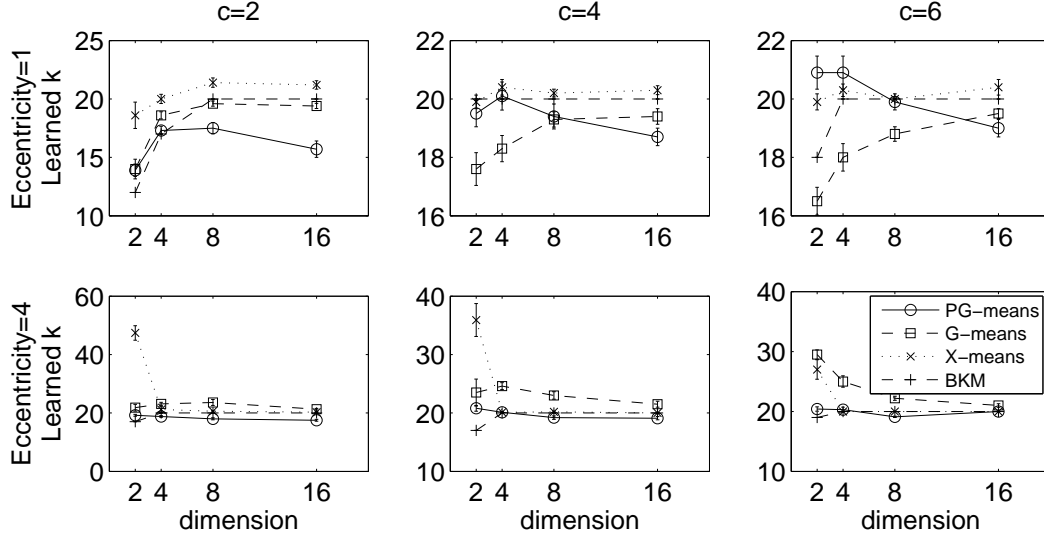

Figure 1: Each point represents the average number of clusters learned for various types of synthetic datasets. The true number of clusters is 20. The error bars denote the standard errors for the experiments (except for BKM, which was run once for each dataset type).

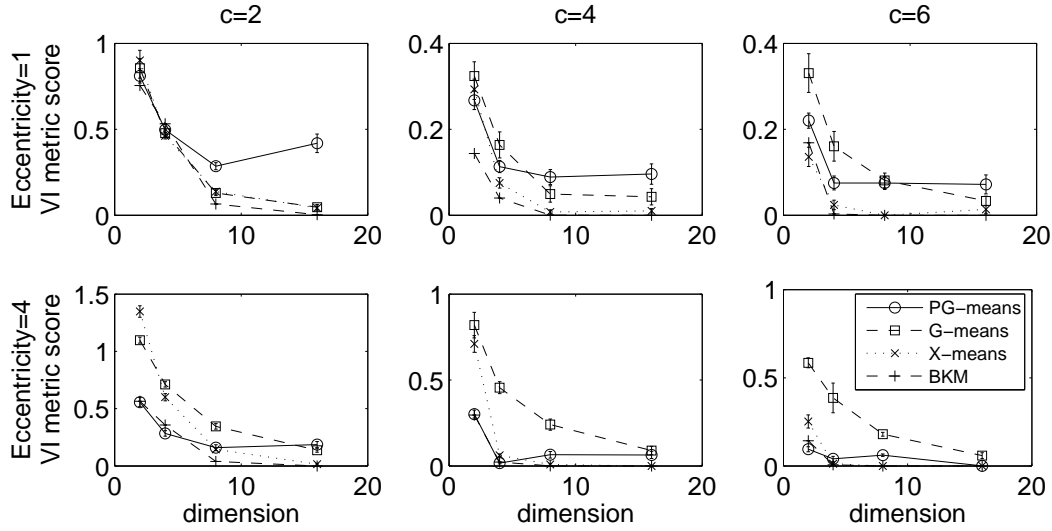

Figure 2: Each point represents the average VI metric comparing the learned clustering to the correct labels for various types of synthetic datasets. Lower values are better. For each algorithm except BKM we provide standard error bars (BKM was run once for each dataset type).

## 4 Experimental evaluation

We perform several experiments on synthetic and real-world datasets to illustrate the utility of PG-means and compare it with G-means, X-means, and Bayesian $k$-means (BKM). For synthetic datasets, we experiment with Gaussian and non-Gaussian data. We use $\alpha = 0.001$ for both PG-means and G-means. For each model, PG-means uses 12 projections and tests, corresponding to an error rate of $\varepsilon < 0.01$ that it incorrectly accepts. All our experiments use MATLAB on Linux 2.4 on a dual-processor dual-hyperthreaded Intel Xeon 3.06 GHz computer with 2 gigabytes of memory.

Figure 1 shows the number of clusters found by running PG-means, G-means, X-means and BKM on many synthetic datasets. Each of these datasets has 4000 points in $d = 2, 4, 8$ and 16 dimensions.

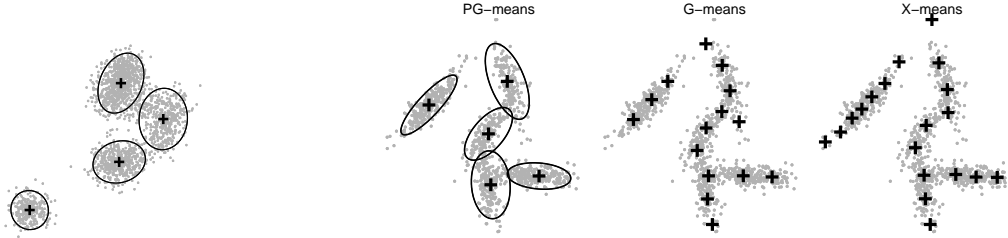

Figure 3: The leftmost dataset has 10 true clusters with significant overlap ($c = 1$). Though PG-means finds only 4 clusters, the model is very reasonable. On the right are the results for PG-means, G-means, and X-means on a dataset with 5 true eccentric and overlapping clusters. PG-means finds the correct model, while the others overfit with 15 and 19 clusters.

All of the data are drawn from a mixture of 20 true Gaussians. The centers of the clusters in each dataset are chosen randomly, and each cluster generates the same number of points. Each Gaussian mixture dataset is specified by the average $c$-separation between each cluster center and its nearest neighbor (either 2, 4 or 6) and each cluster's eccentricity (either 1 or 4). The eccentricity of is defined as $\text{Ecc} = \sqrt{\lambda_{max}/\lambda_{min}}$ where $\lambda_{max}$ and $\lambda_{min}$ are the maximum and minimum eigenvalues of the cluster covariance. An eccentricity of 1 indicates a spherical Gaussian. We generate 10 datasets of each type and run PG-means, G-means and X-means on each, and we run BKM on only one of them due to the running time of BKM. Each algorithm starts with one center, and we do not place an upper-bound on the number of clusters.

It is clear that PG-means performs better than G-means and X-means when the data are eccentric (Ecc=4), especially when the clusters overlap ($c = 2$). In this situation G-means and X-means tend to overestimate the number of clusters. The rightmost plots in Figure 3 further illustrate this overfitting. PG-means is much more stable in its estimate of the number of clusters, unlike G-means and X-means which can dramatically overfit depending on the type of data. BKM generally does very well, but is less efficient than PG-means. For example, on a set of 24 different datasets each having 4000 points from 10 clusters, 2-16 dimensions and varying separations/eccentricities, PG-means was three times faster than BKM.

Figure 1 only gives the information regarding the learned number of clusters, which is not enough to measure the true quality of learned models. In order to better evaluate the approaches, we use Meila's VI (Variation of Information) metric [8] to compare the induced clustering to the true labels. The VI metric is non-negative and lower values are better. It is zero when the two compared clusterings are identical (modulo clusters being relabeled). Figure 2 shows the average VI metric obtained by running PG-means, G-means, X-means, and BKM on the same synthetic datasets as in Figure 1. PG-means does about as well as the other algorithms when the data are spherical and well-separated (see the top-right plot). However, the top-left plot shows that PG-means does not perform as well as G-means, X-means and BKM for spherical and overlapping data. The reason is that two spherical clusters overlap, they can look like a single eccentric cluster. Since PG-means can capture eccentric clusters effectively, it will accept these two overlapped spherical clusters as one cluster. But for the same case, G-means and X-means will probably recognize them as two different clusters. Therefore, although PG-means gives fewer clusters for spherical and overlapping data, the models it learns are reasonable. Figure 3 shows how 10 true overlapping clusters may look like far fewer clusters, and that PG-means can find an appropriate model with only 4 clusters.

High dimensional data of any finite-variance distribution looks more Gaussian when linearly projected to a randomly chosen lower-dimensional space. Projection is a weighted sum of the original dimensions, and the sum of many random variables with finite variance tends to be Gaussian, according to the central limit theorem. Thus PG-means should be useful for high-dimensional data which are not Gaussian. To test this, we perform experiments on high-dimensional non-Gaussian synthetic datasets. These datasets are generated in a similar way of generating our synthetic Gaussian datasets, except that each true cluster has a uniform distribution. Each cluster is not necessarily axis-aligned or square; it is scaled for eccentricity and rotated. Each dataset has 4000 points in 8 dimensions equally distributed among 20 clusters. The eccentricity and $c$-separation values for the datasets are both 4. We run PG-means, G-means and X-means on 10 different datasets and BKM

Table 1: Results for synthetic non-Gaussian data and the handwritten digits dataset. Each non-Gaussian dataset contains 4000 points in 8 dimensions sampled from 20 true clusters each having uniform distribution. The eccentricity and $c$-separation are both 4. We run each algorithm except BKM on ten such datasets, and BKM on one. The digits dataset consists of 10 classes and 9298 examples.

|  | Non-Gaussian datasets (20 true clusters) | | Handwritten digits dataset (10 true classes) | |
| --- | --- | --- | --- | --- |
| Algorithm | Learned $k$ | VI metric | Learned $k$ | VI metric |
| PG-means | $20 \pm 0$ | $0 \pm 0$ | 14 | 2.045 |
| G-means | $42.2 \pm 3.67$ | $0.673 \pm 0.071$ | 48 | 3.174 |
| X-means | $27.7 \pm 1.28$ | $0.355 \pm 0.059$ | 29 | 2.921 |
| BKM | 20 | 0 | 15 | 1.980 |

on one of them. The results are shown in the left part of Table 1. G-means and X-means overfit the non-Gaussian datasets, while PG-means and BKM both perform excellently in the number of clusters learned and in learning the true labels according to the VI metric.

We tested all of these algorithms on the U.S. Postal Service handwritten digits dataset (both the train and test portions, obtained from http://www-stat.stanford.edu/~tibs/ElemStatLearn/data.html). Each example is a grayscale image of a handwritten digit. There are 9298 examples in the dataset, and each example has 256 pixels (16 pixels on a side). The dataset has 10 true classes (digits 0-9). Our goal is to cluster the dataset without knowing the true labels and analyze the result to find out how well PG-means captures the true classes.

We use random linear projection to project the dataset to 16 dimensions and run PG-means, G-means, X-means, and BKM on it. The results are shown in the right side of Table 1. PG-means gives 14 centers, which is closest to the true value. It also obtains nearly the best VI metric score. On the other hand, G-means and X-means find many more classes than the truth, which do not help them score well on the VI metric, and BKM takes over twice as long as PG-means.

## 5 Conclusions and future work

We presented a new algorithm called PG-means for learning the number of Gaussian clusters $k$ in data. Starting with one center, it grows $k$ gradually. For each $k$, it learns a model using Expectation-Maximization. Then it projects both the model and the dataset to one dimension and tests for model fitness with the Kolmogorov-Smirnov test and its own critical values. It performs multiple projections and tests per model, to avoid being fooled by a poorly chosen projection. If the model does not fit well, PG-means adds an additional cluster. This procedure repeats until one model is accepted by all tests. We proved that only a small number of these fast tests are required to have good performance at finding model differences. In the future we will investigate methods of finding better projections for our task. We also hope to develop approximations to the critical values of the KS test on Gaussian mixtures, to avoid the cost of Monte Carlo simulations.

PG-means finds better models than G-means and X-means when the true clusters are eccentric or overlap, especially in low-dimension. On high-dimensional data PG-means also performs very well. PG-means gives far more stable estimates of the number of clusters than the other two methods over many different types of data. Compared with Bayesian $k$-means, we showed that PG-means performs comparably, though PG-means is several times faster in our tests and uses less memory.

Though PG-means looks for general Gaussian clusters, we showed that PG-means works well on high-dimensional non-Gaussian data, due to the central limit theorem and our use of projection. Our techniques would also be applicable as a wrapper around the $k$-means algorithm, which is really just a mixture of spherical Gaussians, or any other mixture of Gaussians with limited covariance. On the real-world handwritten digits dataset PG-means finds a very good clustering with nearly the correct number of classes, and PG-means and BKM are equally close to identifying the original labels among the algorithms we tested.

We believe that the project-and-test procedure that PG-means uses is a useful method for determining fitness of a given mixture of Gaussians. However, the underlying standard EM clustering algorithm dominates the runtime and is difficult to initialize well, which are well-known problems.

The project-and-test framework of PG-means does not depend on EM in any way, and could be wrapped around any other better method of finding a Gaussian mixture.

**Acknowledgements**: We thank Dennis Johnston, Sanjoy Dasgupta, Charles Elkan, and the anonymous reviewers for helpful suggestions. We also thank Dan Pelleg and Ken Kurihara for sending us their source code.

# References

[1] Hirotugu Akaike. A new look at the statistical model identification. *IEEE Transactions on Automatic Control*, 19:716–723, 1974.

[2] Gerard E. Dallal and Leland Wilkinson. An analytic approximation to the distribution of Lilliefors' test for normality. *The American Statistician*, 40:294–296, 1986.

[3] Sanjoy Dasgupta. Experiments with random projection. In *Proceedings of the Sixteenth Conference on Uncertainty in Artificial Intelligence (UAI-2000)*, pages 143–151. Morgan Kaufmann Publishers, 2000.

[4] Greg Hamerly and Charles Elkan. Learning the $k$ in $k$-means. In *Proceedings of the seventeenth annual conference on neural information processing systems (NIPS)*, pages 281–288, 2003.

[5] Robert E. Kass and Larry Wasserman. A reference Bayesian test for nested hypotheses and its relationship to the schwarz criterion. *Journal of the American Statistical Association*, 90(431):928–934, 1995.

[6] Hubert W. Lilliefors. On the Kolmogorov-Smirnov test of normality with mean and variance unknown. *Journal of the American Statistical Association*, 62(318):399–402, 1967.

[7] Frank J. Massey, Jr. The Kolmogorov-Smirnov test for goodness of fit. *Journal of the American Statistical Association*, 46(253):68–78, 1951.

[8] Marina Meila. Comparing clusterings by the variation of information. In *COLT*, pages 173–187, 2003.

[9] Dan Pelleg and Andrew Moore. X-means: Extending k-means with efficient estimation of the number of clusters. In *Proceedings of the 17th International Conf. on Machine Learning*, pages 727–734. Morgan Kaufmann, 2000.

[10] Jorma Rissanen. Modeling by shortest data description. *Automatica*, 14:465–471, 1978.

[11] Peter Sand and Andrew W. Moore. Repairing faulty mixture models using density estimation. In *Proceedings of the 18th International Conf. on Machine Learning*, pages 457–464, 2001.

[12] Gideon Schwarz. Estimating the dimension of a model. *The Annnals of Statistics*, 6(2):461–464, 1978.

[13] Robert Tibshirani, Guenther Walther, and Trevor Hastie. Estimating the number of clusters in a dataset via the Gap statistic. *Journal of the Royal Statistical Society B*, 63:411–423, 2001.

[14] Naonori Ueda and Ryohei Nakano. Deterministic annealing em algorithm. *Neural Networks*, 11(2):271–282, 1998.

[15] Max Welling and Kenichi Kurihara. Bayesian k-means as a 'maximization-expectation' algorithm. In *SIAM conference on Data Mining SDM06*, 2006.
